# Dynamic Behavior of Constrained
# Back–Propagation Networks

Yves    Chauvin[1]
Thomson–CSF, Inc.
630 Hansen Way, Suite 250
Palo Alto, CA. 94304

## ABSTRACT

The learning dynamics of the back–propagation algorithm are investigated when complexity constraints are added to the standard Least Mean Square (LMS) cost function. It is shown that loss of generalization performance due to overtraining can be avoided when using such complexity constraints. Furthermore, "energy," hidden representations and weight distributions are observed and compared during learning. An attempt is made at explaining the results in terms of linear and non–linear effects in relation to the gradient descent learning algorithm.

## 1  INTRODUCTION

It is generally admitted that generalization performance of back–propagation networks (Rumelhart, Hinton & Williams, 1986) will depend on the relative size of the training data and of the trained network. By analogy to curve–fitting and for theoretical considerations, the generalization performance of the network should decrease as the size of the network and the associated number of degrees of freedom increase (Rumelhart, 1987; Denker et al., 1987; Hanson & Pratt, 1989).

This paper examines the dynamics of the standard back–propagation algorithm (BP) and of a constrained back–propagation variation (CBP), designed to adapt the size of the network to the training data base.  The performance, learning dynamics and the representations resulting from the two algorithms are compared.

1.    Also in the Psychology Department, Stanford University, Stanford, CA. 94305

## 2  GENERALIZATION PERFORMANCE

### 2.1  STANDARD BACK-PROPAGATION

In Chauvin (In Press), the generalization performance of a back–propagation net-work was observed for a classification task from spectrograms into phonemic cate-gories (single speaker, 9 phonemes, 10msx16frequencies spectrograms, 63 training patterns, 27 test patterns). This performance was examined as a function of the number of training cycles and of the number of (logistic) hidden units (see also, Morgan & Bourlard, 1989). During early learning, the performance of the network appeared to be basically independent of the number of hidden units (provided a minimal size). However, after prolonged training, performance started to decrease with training at a rate that was a function of the size of the hidden layer. More precisely, from 500 to 10,000 cycles, the generalization performance (in terms of percentage of correctly classified spectrograms) decreased from about 93% to 74% for a 5 hidden unit network and from about 95% to 62% for a 10 hidden unit network. These results confirmed the basic hypothesis proposed in the Introduc-tion but only with a sufficient number of training cycles (overtraining).

### 2.2  CONSTRAINED BACK-PROPAGATION

Several constraints have been proposed to "adapt" the size of the trained network to the training data. These constraints can act directly on the weights, or on the net input or activation of the hidden units (Rumelhart, 1987; Chauvin, 1987, 1989, In Press; Hanson & Pratt, 1989; Ji, Snapp & Psaltis, 1989; Ishikawa, 1989; Golden and Rumelhart, 1989). The complete cost function adopted in Chauvin (In Press) for the speech labeling task was the following:

$$C = aE_r + \beta E_n + \gamma W = a \sum_{ip}^{OP} (t_{ip} - o_{ip})^2 + \beta \sum_{ip}^{HP} \frac{o_{ip}^2}{1 + o_{ip}^2} + \gamma \sum_{ij}^{W} \frac{w_{ij}^2}{1 + w_{ij}^2} \qquad [1]$$

$E_r$ is the usual LMS error computed at the output layer, $E_n$ is a function of the squared activations of the hidden units and $W$ is a function of the squared weights throughout the network. This constrained back–propagation (CBP) algorithm ba-sically eliminated the overtraining effect: the resulting generalization performance remained constant (about 95%) throughout the complete training period, indepen-dently of the original network size.

## 3  ERROR AND ENERGY DYNAMICS

Using the same speech labeling task as in Chauvin (In Press), the dynamics of the global variables of the network defined in Equation 1 ($E_r$, $E_n$, and $W$) were observed during training of a network with 5 hidden units. Figure 1 represents the error and energy dynamics for the standard (BP) and the constrained back–propa-gation algorithm (CBP). For BP and CBP, the error on the training patterns kept

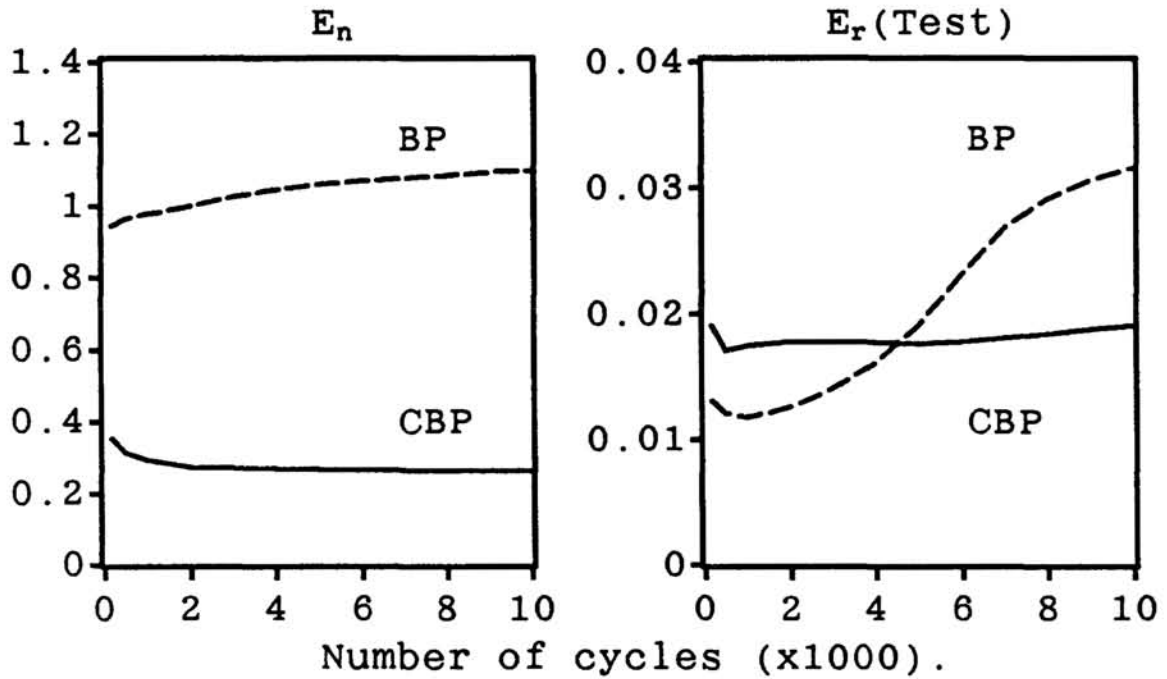

Figure 1. "Energy" (left) and generalization error – LMS averaged over the test patterns and output units – (right) when using the standard (BP) or the constrainted (CBP) back–propagation algorithm during a typical run.

decreasing during the entire training period (more slowly for CBP). The W dynamics over the entire network were similar for BP and CBP (but the distributions were different, see below).

## 3.1 STANDARD BACK–PROPAGATION

As shown in Figure 1, the "energy" $E_n$ (Equation 1) of the hidden layer slightly increases during the entire learning period, long after the minimum was reached for the test error (around 200 cycles). This "energy" reaches a plateau after long overtraining, around 10,000 cycles. The generalization error reaches a minimum and later increases as training continues, also slowly reaching a plateau around 10,000 cycles.

## 3.2 CONSTRAINED BACK–PROPAGATION

With CBP, the "energy" decreases to a much lower level during early learning and remains about constant throughout the complete training period. The error quickly decreases during early learning *and* remains about constant during the rest of the training period, apparently stabilized by the energy and weight constraints given in Equation 1.

# 4  REPRESENTATION

The hidden unit activations and weights of the networks were examined after learning, using BP or CBP. A hidden unit was considered "dead" when its contribution to any output unit (computed as the product of its activation times the corresponding outgoing weight) was at least 50 times smaller than the total contribution from all hidden units, over the entire set of input patterns.

## 4.1  STANDARD BACK-PROPAGATION

As also observed by Hanson et al. (1989), standard back-propagation usually makes use of most or all hidden units: the representation of the input patterns is well distributed over the entire set of hidden units, even if the network is oversized for the task. The exact representation depends on the initial weights.

## 4.2  CONSTRAINED BACK-PROPAGATION

Using the constraints described in Equation 1, the hidden layer was reduced to 2 or 3 hidden units for all the observed runs (2 hidden units corresponds to the minimal size network necessary to solve the task). All the other units were actually "killed" during learning, independently of the size of the original network (from 4 to 11 units in the simulations). Both the constraints on the hidden unit activations ($E_n$) and on the weights ($W$) contribute to this reduction.

Figure 2 represents an example of the resulting weights from the input layer to a remaining hidden unit. As we can see, a few weights ended up dominating the entire set: they actually "picked up" a characteristic of the input spectrograms that allow the disctinction between two phoneme categories (this phenomenon was also predicted and observed by Rumelhart, 1989). In this case, the weights "picked up" the 10th and 14th frequency components of the spectrograms, both present during the 5th time interval. The characteristics of the spectrum make the corresponding hidden unit especially responsive to the [G] phoneme. The specific non-linear $W$ constraint on the input-to-hidden weights used by CBP forced that hidden unit to acquire a very local receptor field. Note that this was not always observed in the simulations. Some hidden units acquired broad receptor fields with weights distributed over the entire spectrogram (as it is always the case with standard BP). No statistical comparison was made to compute the relative ratio of local to distributed units, which probably depends on the exact form of the reduction constraint used in CBP.

# 5  INTERPRETATION OF RESULTS

We observed that the occurrence of overfitting effects depends both on the size of the network *and* on the number of training cycles. At this point, a better theoretical understanding of the back-propagation learning dynamics would be useful to explain this dependency (Chauvin, In Preparation). This section presents an informal interpretation of the results in terms of linear and non-linear phenomena.

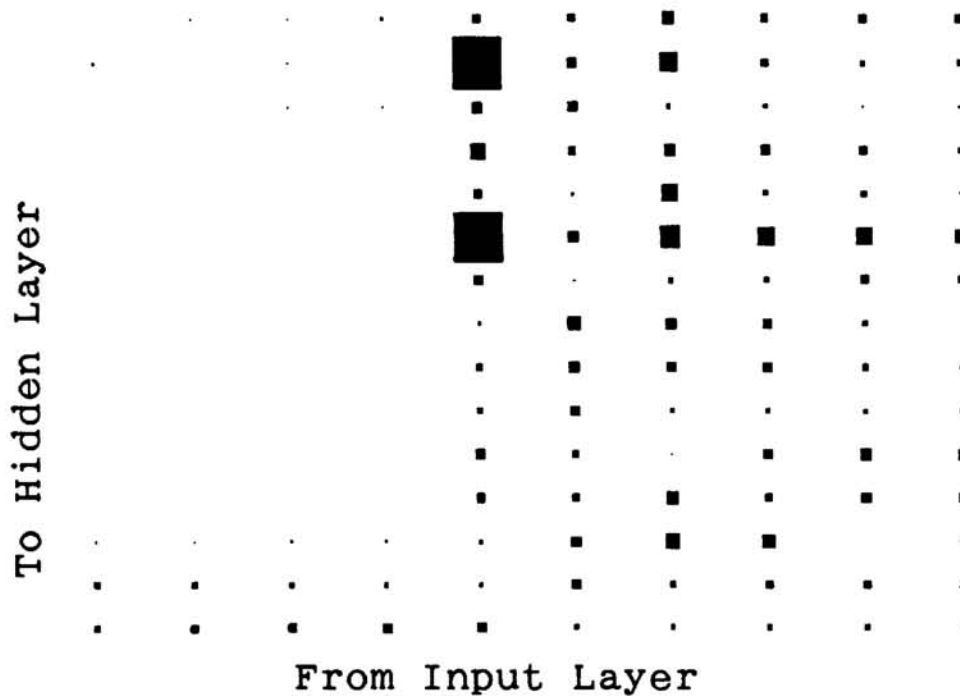

**Figure 2.** Typical fan-in weights after learning from the input layer to a hidden unit using the constrained back-propagation algorithm.

## 5.1 LINEAR PHENOMENA

These linear phenomena might be due to probable correlations between sample plus observation noise at the input level and the desired classification at the output level. The gradient descent learning rule should eventually make use of these correlations to decrease the LMS error. However, these correlations are specific to the used training data set and should have a negative impact on the performance of the network on a testing data set. Figure 3 represents the generalization performance of linear networks with 1 and 7 hidden units (averaged over 5 runs) for the speech labeling task described above. As predicted, we can see that overtraining effects are actually generated by linear networks (as they would with a one-step algorithm; e.g., Vallet et al., 1989). Interestingly, they occur even when the size of the network is minimum. These effects should obviously decrease by increasing the size of the training data set (therefore reducing the effect of sample and observation noise).

## 5.2 NON-LINEAR PHENOMENA

The second type of effect is non-linear. This is illustrated in Chauvin (In Press) with a curve-fitting problem. In the first problem, a non-linear back-propagation network (1 input unit, 1 output unit, 2 layers of 20 hidden units) is trained to fit a function composed of two linear segments separated by a discontinuity. The mapping realized by the network over the entire interval is observed as a function of the number of training cycles. It appears that the interpolated fit reaches a minimum

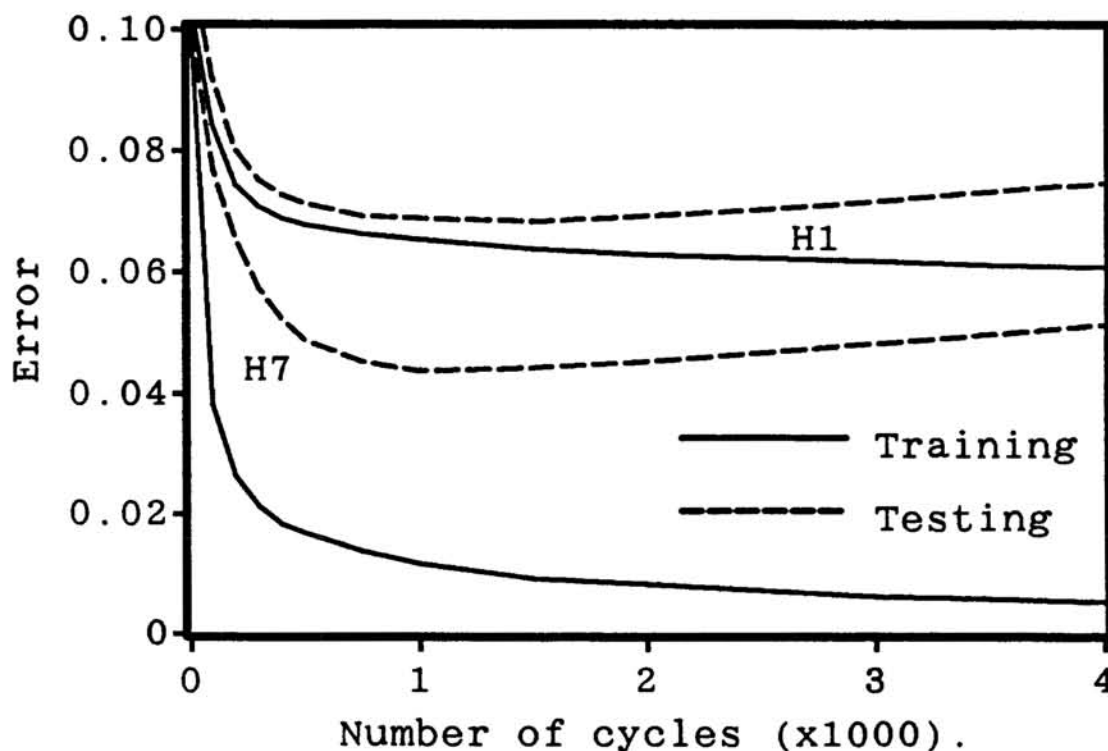

**Figure 3.** LMS error for the training and test data sets of a speech labeling task as a function of the number of training cycles. A one hidden and a 7 hidden unit linear network are considered.

and gets worse with the number of training cycles *and* with the size of the sample training set around the discontinuity.

This phenomenon is evocative of an effect in interpolation theory known as the Runge effect (Steffenssen, 1950). In this case, a "well–behaved" bell–like function, $f(x) = 1/(1 + x^2)$, uniformly sampled $n+1$ times over a $[-D, +D]$ interval, is fitted with a polynomial of degree $n$. Runge showed that over the considered interval, the maximum distance between the fitted function and the fitting polynomial goes to infinity as $n$ increases. Note that in theory, there is no overfitting since the number of degree of freedoms associated with the polynomial matches the number of data points. However, the interpolation "overfitting effect" actually increases with the sampling data set, that is with the increased accuracy in the description of the fitted function. (Runge also showed that the effect may disappear by changing the size of the sampled interval or the distribution of the sampling data points.)

We can notice that in the piecewise linear example, a linear network would have computed a linear mapping using only two degrees of freedom (the problem is then equivalent to one–dimensional linear regression). With a non–linear network, simulations show that the network actually computes the desired mapping by slowly

fitting higher and higher "frequency components" present in the desired mapping (reminiscent of the Gibb's phenomenon observed with successive Fourier series approximations of a square wave; e.g., Sommerfeld, 1949). The discontinuity, considered as a singular point with high frequency components, is fitted during later stages of learning. Increasing the number of sampling points around the disconti- nuilty generates an effect similar to the Runge effect with overtraining. In this sense, the notion of degrees of freedom in non–linear neural networks is not only a function of the network architecture – the "capacity" of the network – and of the non–linearities of the fitted function but also of the learning algorithm (gradient descent), which *gradually* "adjusts" the "capacity" of the network to fit the non- linearities required by the desired function.

A practical classification task might generate not only linear overtraining effects due to sample and observation noise but also non–linear effects if a continuous input variable (such as a frequency component in the speech example) has to be classified in two different bins. It is also easy to imagine that noise may generate non–linear effects. At this stage, the non–linear effects involved in back–propaga- tion networks composed of logistic hidden units are poorly understood. In general, both effects will probably occur in non–linear networks and might be difficult to assess. However, because of the gradient descent procedure, both effects seem to depend on the amount of training relative to the capacity of the network. The use of complexity constraints acting on the complexity of the network seems to consti- tute a promising solution to the overtraining problem in both the linear and non–li- near cases.

## Acknowledgements

I am greatful to Pierre Baldi, Fred Fisher, Matt Franklin, Richard Golden, Julie Holmes, Erik Marcade, Yoshiro Miyata, David Rumelhart and Charlie Schley for helpful comments.

## References

Chauvin, Y. (1987). Generalization as a function of the number of hidden units in back–propagation networks. *Unpublished Manuscript*. University of California, San Diego, CA.

Chauvin, Y. (1989). A back–propagation algorithm with optimal use of the hidden units. In D. Touretzky (Ed.), *Advances in Neural Information Processing Systems 1*. Palo Alto, CA: Morgan Kaufman.

Chauvin, Y. (In Press). Generalization performance of back–propagation networks. *Proceedings of the 1990 European conference on Signal Processing (Eurasip)*. Springer–Verlag.

Chauvin, Y. (In Preparation). Generalization performance of LMS trained linear networks.

Denker, J. S., Schwartz, D. B., Wittner, B. S., Solla, S. A., Howard, R. E., Jackel, L. D., & Hopfield, J. J. (1987). Automatic learning, rule extraction, and generalization. *Complex systems, 1*, 877–922.

Golden, R.M., & Rumelhart, D.E. (1989). *Improving generalization in multi-layer networks through weight decay and derivative minimization.* Unpublished Manuscript. Stanford University, Palo Alto, CA.

Hanson, S. J. & Pratt, L. P. (1989). Comparing biases for minimal network construction with back-propagation. In D. Touretzky (Ed.), *Advances in Neural Information Processing Systems 1*. Palo Alto, CA: Morgan Kaufman.

Ishikawa M. (1989). A structural learning algorithm with forgetting of weight link weights. *Proceedings of the IJCNN International Joint Conference on Neural Networks, II*, 626. Washington D.C., June 18–22, 1989.

Ji, C., Snapp R. & Psaltis D. (1989). Generalizing smoothness constraints from discrete samples. *Unpublished Manuscript*. Department of Electrical Engineering. California Institute of Technology, CA.

Morgan, N. & Bourlard, H. (1989). *Generalization and parameter estimation in feedforward nets: some experiments.* Paper presented at the Snowbird Conference on Neural Networks, Utah.

Rumelhart, D. E., Hinton G. E., Williams R. J. (1986). Learning internal representations by error propagation. In D. E. Rumelhart & J. L. McClelland (Eds.) *Parallel Distributed Processing: Explorations in the Microstructures of Cognition (Vol. I)*. Cambridge, MA: MIT Press.

Rumelhart, D. E. (1987). Talk given at Stanford University, CA.

Rumelhart, D. E. (1989). Personal Communication.

Sommerfeld, A. (1949). *Partial differential equations in physics.* (Vol. VI). Academic Press: New York, NY.

Steffenssen, J. F. (1950). *Interpolation.* Chelsea: New York, NY.

Vallet, F., Cailton, J.-G. & Refregier P. (1989). Solving the problem of overfitting of the pseudo-inverse solution for classification learning. *Proceedings of the IJCNN Conference, II*, 443–450. Washington D.C., June 18–22, 1989.
